# Redundancy and Dimensionality Reduction in Sparse-Distributed Representations of Natural Objects in Terms of Their Local Features

**Penio S. Penev***
Laboratory of Computational Neuroscience
The Rockefeller University
1230 York Avenue, New York, NY 10021
*penev@rockefeller.edu*   *http://venezia.rockefeller.edu/*

## Abstract

Low-dimensional representations are key to solving problems in high-level vision, such as face compression and recognition. Factorial coding strategies for reducing the redundancy present in natural images on the basis of their second-order statistics have been successful in accounting for both psychophysical and neurophysiological properties of early vision. Class-specific representations are presumably formed later, at the higher-level stages of cortical processing. Here we show that when retinotopic factorial codes are derived for ensembles of natural objects, such as human faces, not only redundancy, but also dimensionality is reduced. We also show that objects are built from parts in a non-Gaussian fashion which allows these local-feature codes to have dimensionalities that are substantially lower than the respective Nyquist sampling rates.

## 1   Introduction

Sensory systems must take advantage of the statistical structure of their inputs in order to process them efficiently, both to suppress noise and to generate compact representations of seemingly complex data. *Redundancy reduction* has been proposed as a design principle for such systems (Barlow, 1961); in the context of Information Theory (Shannon, 1948), it leads to *factorial codes* (Barlow et al., 1989; Linsker, 1988). When only the second-order statistics are available for a given sensory ensemble, the maximum entropy initial assumption (Jaynes, 1982) leads to a multi-dimensional Gaussian model of the probability density; then, the *Karhunen-Loève Transform (KLT)* provides a family of equally efficient factorial codes. In the context of the ensemble of natural images, with a specific model for the noise, these codes have been able to account quantitatively for the contrast sensitivity of human subjects in all signal-to-noise regimes (Atick and Redlich, 1992). Moreover, when the receptive fields are constrained to have *retinotopic organization,* their circularly symmetric, center-surround opponent structure is recovered (Atick and Redlich, 1992).

Although redundancy can be reduced in the ensemble of natural images, because its spectrum obeys a power law (Ruderman and Bialek, 1994), there is no natural cutoff, and the

*dimensionality* of the "retinal" code is the same as that of the input. This situation is not typical. When KLT representations are derived for ensembles of natural objects, such as human faces (Sirovich and Kirby, 1987), the factorial codes in the resulting families are naturally low-dimensional (Penev, 1998; Penev and Sirovich, 2000). Moreover, when a retinotopic organization is imposed, in a procedure called *Local Feature Analysis (LFA)*, the resulting feed-forward receptive fields are a dense set of detectors for the local features from which the objects are built (Penev and Atick, 1996). LFA has also been used to derive local features for the natural-objects ensembles of: 3D surfaces of human heads (Penev and Atick, 1996), and 2D images of pedestrians (Poggio and Girosi, 1998).

Parts-based representations of object classes, including faces, have been recently derived by *Non-negative Matrix Factorization (NMF)* (Lee and Seung, 1999), "biologically" motivated by the hypothesis that neural systems are incapable of representing negative values. As has already been pointed out (Mel, 1999), this hypothesis is incompatible with a wealth of reliably documented neural phenomena, such as center-surround receptive field organization, excitation and inhibition, and ON/OFF visual-pathway processing, among others.

Here we demonstrate that when parts-based representations of natural objects are derived by redundancy reduction constrained by retinotopy (Penev and Atick, 1996), the resulting sparse-distributed, local-feature representations not only are factorial, but also are of dimensionalities substantially lower than the respective Nyquist sampling rates.

## 2   Compact Global Factorial Codes of Natural Objects

A properly registered and normalized object will be represented by the receptor readout values $\phi(\mathbf{x})$, where $\{\mathbf{x}\}$ is a grid that contains $V$ receptors. An *ensemble* of $T$ objects will be denoted by $\{\phi^t(\mathbf{x})\}_{t \in T}$.[1] Briefly (see, *e.g.*, Sirovich and Kirby, 1987, for details), when $T > V$, its *Karhunen-Loève Transform (KLT)* representation is given by

$$\phi^t(\mathbf{x}) = \sum_{r=1}^{V} a_r^t \sigma_r \psi_r(\mathbf{x}) \tag{1}$$

where $\{\sigma_r^2\}$ (arranged in non-increasing order) is the *eigenspectrum* of the spatial and temporal correlation matrices, and $\{\psi_r(\mathbf{x})\}$ and $\{a_r^t\}$ are their respective *orthonormal* eigenvectors. The KLT representation of an arbitrary, possibly out-of-sample, object $\phi(\mathbf{x})$ is given by the joint activation

$$a_r = \frac{1}{V} \sum_{\mathbf{x}} \sigma_r^{-1} \psi_r(\mathbf{x}) \phi(\mathbf{x}) \tag{2}$$

of the set of *global analysis filters* $\{\sigma_r^{-1}\psi_r(\mathbf{x})\}$, which are indexed with $r$, and whose outputs, $\{a_r\}$, are decorrelated.[2] In the context of the ensemble of natural images, the "whitening" by the factor $\sigma_r^{-1}$ has been found to account for the contrast sensitivity of human subjects (Atick and Redlich, 1992). When the output dimensionality is set to $N < V$, the *reconstruction*—optimal in the amount of preserved signal power—and the respective *error* utilize the *global synthesis filters* $\{\sigma_r \psi_r(\mathbf{x})\}$, and are given by

$$\phi_N^{rec} = \sum_{r=1}^{N} a_r \sigma_r \psi_r \quad \text{and} \quad \phi_N^{err} = \phi - \phi_N^{rec}. \tag{3}$$

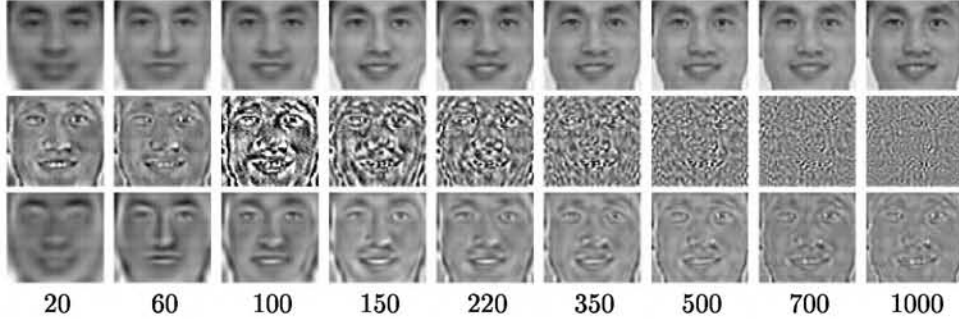

| 20 | 60 | 100 | 150 | 220 | 350 | 500 | 700 | 1000 |

Figure 1: Successive reconstructions, errors, and local entropy densities. For the indicated global dimensionalities, $N$, the reconstructions $\phi_N^{rec}$ (3) of an *out-of-sample* example are shown in the top row, and the respective residual errors, $\phi_N^{err}$, in the middle row (the first two errors are amplified $5\times$ and the rest—$20\times$). The respective entropy densities $O_N$ (5), are shown in the bottom, low-pass filtered with $F_{r,N} = \sigma_r^2/(\sigma_r^2 + \sigma_N{}^2)$ (cf. Fig. 3), and scaled adaptively at each $N$ to fill the available dynamic range.

With the standard *multidimensional Gaussian* model for the probability density $\mathcal{P}[\phi]$ (Moghaddam and Pentland, 1997; Penev, 1998), the *information content* of the reconstruction (3)—equal to the *optimal-code length* (Shannon, 1948; Barlow, 1961)—is

$$-\log \mathcal{P}[\phi_N^{rec}] \propto \sum_{r=1}^{N} |a_r|^2. \tag{4}$$

Because of the normalization by $\sigma_r$ in (2), all KLT coefficients have *unit variance* (1); the model (4) is *spherically symmetric,* and all filters contribute equally to the entropy of the code. What criterion, then, could guide dimensionality reduction?

Following (Atick and Redlich, 1992), when noise is taken into account, $N \approx 400$ has been found as an estimate of the global dimensionality for the ensemble frontal-pose faces (Penev and Sirovich, 2000). This conclusion is reinforced by the perceptual quality of the successive reconstructions and errors, shown in Fig. 1—the face-specific information crosses over from the error to the reconstruction at $N \approx 400$, but not much earlier.

## 3   Representation of Objects in Terms of Local Features

It was shown in Section 2 that when redundancy reduction on the basis of the second-order statistics is applied to ensembles of natural objects, the resulting factorial code is compact (low dimensional), in contrast with the "retinal" code, which preserves the dimensionality of the input (Atick and Redlich, 1992). Also, the filters in the beginning of the hierarchy (Fig. 2) correspond to intuitively understandable sources of variability. Nevertheless, this compact code has some problems. The learned receptive fields, shown in Fig. 2, are *global,* in contrast with the local, retinotopic organization of sensory processing, found throughout most of the visual system. Moreover, although the eigenmodes in the regime $r \in [100, 400]$ are clearly necessary to preserve the object-specific information (Fig. 1), their respective global filters (Fig. 2) are ripply, non-intuitive, and resemble the hierarchy of sine/cosine modes of the translationally invariant ensemble of natural images.

In order to cope with these problems in the context of object ensembles, analogously to the local factorial retinal code (Atick and Redlich, 1992), *Local Feature Analysis (LFA)* has been developed (Penev and Atick, 1996; Penev, 1998). LFA uses a set of *local analysis fil-*

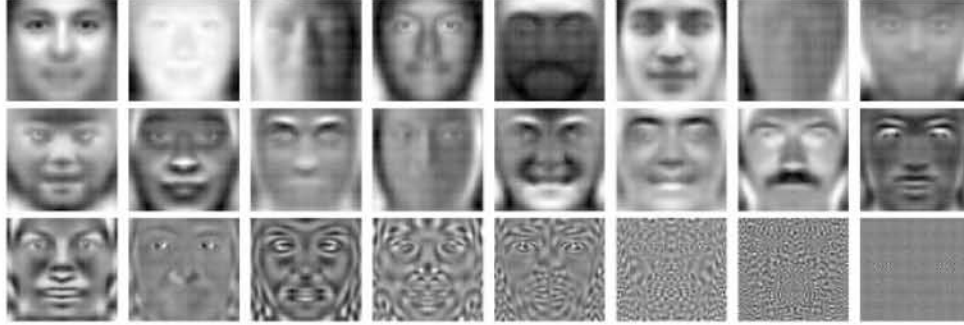

Figure 2: The basis-vector hierarchy of the global factorial code. Shown are the first 14 eigenvectors, and the ones with indices: 21, 41; and 60, 94, 155, 250, 500, 1000, 2000, 3840 (bottom row).

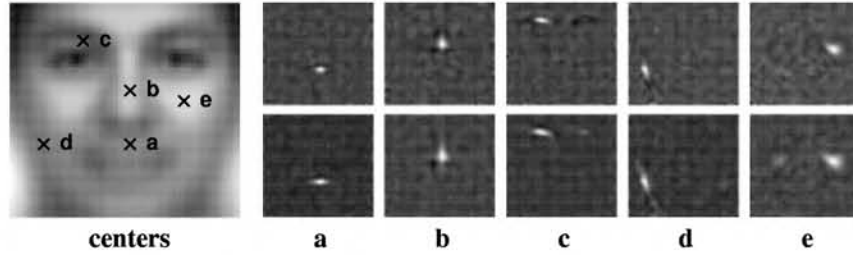

| centers | a | b | c | d | e |

Figure 3: Local feature detectors and residual correlations of their outputs. *centers:* The typical face ($\psi_1$, Fig. 2) is marked with the central positions of five of the feature detectors. *a—e:* For those choises of $\mathbf{x}_\alpha$, the local filters $K(\mathbf{x}_\alpha, \mathbf{y})$ (6) are shown in the top row, and the residual correlations of their respective outputs with the outputs of all the rest, $P(\mathbf{x}_\alpha, \mathbf{y})$ (9), in the bottom. In principle, the cutoff at $r = N$, which effectively implements a low-pass filter, should not be as sharp as in (6)—it has been shown that the human contrast sensitivity is described well by a smooth cutoff of the type $F_r = \sigma_r^2/(\sigma_r^2 + n^2)$, where $n^2$ is a mearure of the effective noise power (Atick and Redlich, 1992). For this figure, $K(\mathbf{x}, \mathbf{y}) = \sum_{r=1}^{N} \psi_r(\mathbf{x}) \frac{F_r}{\sigma_r} \psi_r(\mathbf{y})$, with $N = 400$ and $n = \sigma_{400}$.

*ters,* $K(\mathbf{x}, \mathbf{y})$, whose outputs are topographically indexed with the grid variable $\mathbf{x}$ (cf. eq. 2)

$$O(\mathbf{x}) \triangleq \frac{1}{V} \sum_{\mathbf{y}} K(\mathbf{x}, \mathbf{y})\phi(\mathbf{y}) \qquad (5)$$

and are as decorrelated as possible. For a given dimensionality, or width of the band, of the compact code, $N$, maximal decorrelation can be achieved with $K(\mathbf{x}, \mathbf{y}) = K_N^{(1)}(\mathbf{x}, \mathbf{y})$ from the following topographic family of kernels

$$K_N^{(n)}(\mathbf{x}, \mathbf{y}) \triangleq \sum_{r=1}^{N} \psi_r(\mathbf{x}) \sigma_r^{-n} \psi_r(\mathbf{y}). \qquad (6)$$

For the ensemble of natural scenes, which is translationally and rotationally invariant, the local filters (6) are center-surround receptive fields (Atick and Redlich, 1992). For object ensembles, the process of construction—*categorization*—breaks a number of symmetries and shifts the higher-order statistics into second-order, where they are conveniently exposed to robust estimation and, subsequently, to redundancy reduction. The resulting *local receptive fields,* some of which are shown in the top row of Fig. 3, turn out to be *feature detectors* that are optimally tuned to the structures that appear at their respective centers. Although the local factorial code does not exhibit the problems discussed earlier, it has

representational properties that are equivalent to those of the global factorial code. The reconstruction and error are identical, but now utilize the *local synthesis filers* $K_N^{(-1)}$ (6)

$$\phi_N^{rec}(\mathbf{x}) = \sum_{r=1}^{N} a_r \sigma_r \psi_r(\mathbf{x}) = \frac{1}{V} \sum_{\mathbf{y}} K_N^{(-1)}(\mathbf{x}, \mathbf{y}) O(\mathbf{y}) \tag{7}$$

and the information (4) is expressed in terms of $O(\mathbf{x})$, which therefore provides the *local information density*

$$-\log \mathcal{P}[\phi_N^{rec}] \propto \sum_{r=1}^{N} |a_r|^2 = \frac{1}{V} \sum_{\mathbf{x}} |O_N(\mathbf{x})|^2. \tag{8}$$

## 4  Greedy Sparsification of the Smooth Local Information Density

In the case of natural images, $N = V$, and the outputs of the local filters are completely decorrelated (Atick and Redlich, 1992). For natural objects, the code is low-dimensional ($N < V$), and *residual correlations,* some shown in the bottom row of Fig. 3, are unavoidable; they are generally given by the projector to the subband

$$P_N(\mathbf{x}, \mathbf{y}) \triangleq \frac{1}{T} \sum_{t} O_N^t(\mathbf{x}) O_N^t(\mathbf{y}) \equiv K_N^{(0)}(\mathbf{x}, \mathbf{y}) \tag{9}$$

and are as close to $\delta(\mathbf{x}, \mathbf{y})$ as possible (Penev and Atick, 1996). The *smoothness* of the local information density is controlled by the width of the band, as shown in Fig. 1. Since $O(\mathbf{x})$ is band limited, it can generally be reconstructed exactly from a *subsampling* over a limited set of grid points $\mathcal{M} \triangleq \{\mathbf{x}_m\}$, from the $|\mathcal{M}|$ variables $\{O_m \triangleq O(\mathbf{x}_m)\}_{\mathbf{x}_m \in \mathcal{M}}$, as long as this density is critically sampled ($|\mathcal{M}| = N$). When $|\mathcal{M}| < N$, the *maximum-likelihood interpolation* in the context of the probability model (8) is given by

$$O^{rec}(\mathbf{x}) = \sum_{m=1}^{|\mathcal{M}|} O_m a_m(\mathbf{x}) \quad \text{with} \quad a_m(\mathbf{x}) = \sum_{n=1}^{|\mathcal{M}|} \mathbf{Q}^{-1}{}_{mn} P_n(\mathbf{x}) \tag{10}$$

where $P_m(\mathbf{x}) \triangleq P(\mathbf{x}_m, \mathbf{x})$, and $\mathbf{Q} \triangleq \mathbf{P}|_{\mathcal{M}}$ is the *restriction* of $\mathbf{P}$ on the set of reference points, with $\mathbf{Q}_{nm} = P_n(\mathbf{x}_m)$ (Penev, 1998). When $O(\mathbf{x})$ is critically sampled ($|\mathcal{M}| = N$) on a regular grid, $V \to \infty$, and the eigenmodes (1) are sines and cosines, then (10) is the familiar Nyquist interpolation formula. In order to improve numerical stability, irregular subsampling has been proposed (Penev and Atick, 1996), by a data-driven greedy algorithm that successively enlarges the support of the subsampling at the $n$-th step, $\mathcal{M}^{(n)}$, by optimizing for the *residual entropy* error, $\|O_n^{err}(\mathbf{x})\|^2 = \|O(\mathbf{x}) - O_n^{rec}(\mathbf{x})\|^2$.

The LFA code is *sparse*. In a recurrent neural-network implementation (Penev, 1998) the *dense* output $O(\mathbf{x})$ of the feed-forward receptive fields, $K(\mathbf{x}, \mathbf{y})$, has been interpreted as *sub-threshold* activation, which is predictively suppressed through lateral inhibition with weights $P_m(\mathbf{x})$, by the set of active units, at $\{\mathbf{x}_m\}$.[3]

## 5  Dimensionality Reduction Beyond the Nyquist Sampling Rate

The efficient allocation of resources by the greedy sparsification is evident in Fig. 4A–B; the most prominent features are picked up first (Fig. 4A), and only a handful of active units are used to describe each individual local feature (Fig. 4B). Moreover, when the dimensionality of the representation is constrained, evidently from Fig. 4C–F, the sparse local code has a much better perceptual quality of the reconstruction than the compact global one.

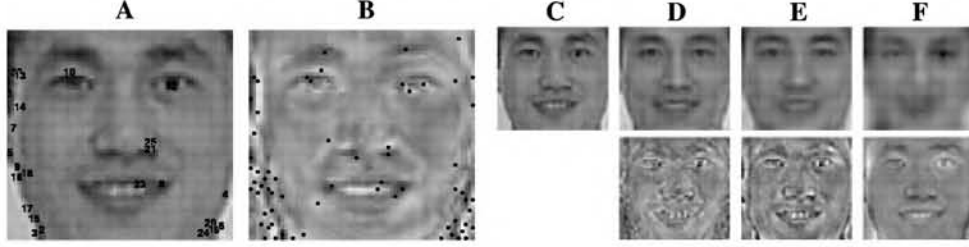

Figure 4: Efficiency of the sparse allocation of resources. $(A)$: the locations of the first 25 active units, $\mathcal{M}^{(25)}$, of the sparsification with $N = 220$, $n = \sigma_{400}$ (see Fig. 3), of the example in Fig. 1 and in $(C)$, are overlaid on $\phi(\mathbf{x})$ and numbered sequentially. $(B)$: the locations of the active units in $\mathcal{M}^{(64)}$ are overlaid on $O(\mathbf{x})$. For $\phi(\mathbf{x})$ in $(C)$ (cf. Fig. 1), reconstructions with a fixed dimensionality, 64, of its deviation from the typical face ($\psi_1$ in Fig. 2), are shown in the top row of $(D, E, F)$, and the respective errors, in the bottom row. $(D)$: reconstruction from the sparsification $\{O(\mathbf{x}_m)\}_{\mathbf{x}_m \in \mathcal{M}}$ (10) with $\mathcal{M} = \mathcal{M}^{(64)}$ from $(B)$. $(E)$: reconstruction from the first 64 global coefficients (3), $N = 64$. $(F)$: reconstruction from a subsampling of $\phi(\mathbf{x})$ on a regular $8 \times 8$ grid (64 samples). The errors in $(D)$ and $(E)$ are magnified $5\times$; in $(F)$, $1\times$.

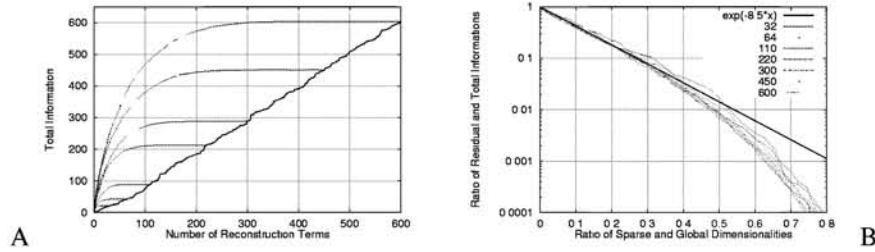

Figure 5: The relationship between the dimensionalities of the global and the local factorial codes. The entropy of the KLT reconstruction (8) for the out-of-sample example (cf. Fig. 1) is plotted in $(A)$ with a solid line as a function of the global dimensionality, $N$. The entropies of the LFA reconstructions (10) are shown with dashed lines parametrically of the number of active units $|\mathcal{M}|$ for $N \in \{600, 450, 300, 220, 110, 64, 32\}$, from top to bottom, respectively. The ratios of the residual, $\|O_n^{err}\|^2$, and the total, $\|O\|^2$ (8), information are plotted in $(B)$ with dashed lines parametrically of $|\mathcal{M}|/N$, for the same values of $N$; a true exponential dependence is plotted with a solid line.

This is an interesting observation. Although the global code is optimal in the amount of captured *energy*, the greedy sparsification optimizes the amount of captured *information*, which has been shown to be the biologically relevant measure, at least in the retinal case (Atick and Redlich, 1992). In order to quantify the relationship between the local dimensionality of the representation and the amount of information it captures, *rate-distortion curves* are shown in Fig. 5. As expected (4), each degree of freedom in the global code contributes approximately equally to the information content. On the other hand, the first few local terms in (10) pull off a sizeable fraction of the total information, with only a modest increase thereafter (Fig. 5$A$). In all regimes for $N$, the residual information decreases approximately exponentially with increasing dimensionality ratio $|\mathcal{M}|/N$ (Fig. 5$B$); 90% of the information is contained in a representation with local dimensionality, 25%–30% of the respective global one; 99%, with 45%–50%. This exponential decrease has been shown to be incompatible with the expectation based on the Gaussian (4), or any other spherical, assumption (Penev, 1999). Hence, the LFA representation, by learning the building blocks of natural objects—the local features—reduces not only redundancy, but also dimensionality. Because LFA captures aspects of the sparse, non-Gaussian structure of natural-object ensembles, it preserves practically all of the information, while allocating resources substantially below the Nyquist sampling rate.

# 6 Discussion

Here we have shown that, for ensembles of natural objects, with low-dimensional global factorial representations, sparsification of the local information density allows under-sampling which results in a substantial additional dimensionality reduction. Although more general ensembles, such as those of natural scenes and natural sound, have full-dimensional global representations, the sensory processing of both visual and auditory signals happens in a multi-scale, bandpass fashion. Preliminary results (Penev and Iordanov, 1999) suggest that sparsification within the subbands is possible beyond the respective Nyquist rate; hence, when the sparse dimensionalities of the subbands are added together, the result is aggregate dimensionality reduction, already at the initial stages of sensory processing.

## Acknowledgments

The major part of this research was made possible by the William O'Baker Fellowship, so generously extended to, and gratefully accepted by, the author. He is also indebted to M. J. Feigenbaum for his hospitality and support; to MJF and A. J. Libchaber, for the encouraging and enlightening discussions, scientific and otherwise; to R. M. Shapley, for asking the questions that led to Fig. 5; to J. J. Atick, B. W. Knight, A. G. Dimitrov, L. Sirovich, J. D. Victor, E. Kaplan, L. G. Iordanov, E. P. Simoncelli, G. N. Reeke, J. E. Cohen, B. Klejn, A. Oppenheim, and A. P. Blicher for fruitful discussions.

## Footnotes

*Present address: NEC Research Institute, 4 Independence Way, Princeton, NJ 08550

[1]For the illustrations in this study, $2\tilde{T} = T = 11254$ frontal-pose facial images were registered and normalized to a grid with $V = 64 \times 60 = 3840$ pixels as previously described (Penev and Sirovich, 2000).

[2]This is certainly true for in-sample objects, since $\{a_r^t\}$ are orthonormal (1). For out-of-sample objects, there is always the issue whether the size of the training sample, $T$, is sufficient to ensure proper generalization. The current ensemble has been found to generalize well in the regime for $r$ that is explored here (Penev and Sirovich, 2000).

[3] This type of sparseness is not to be confused with "high kurtosis of the output distribution;" in LFA, the non-active units are completely shut down, rather than "only weakly activated."

## References

Atick, J. J. and A. N. Redlich (1992). What does the retina know about natural scenes? *Neural Comput.* 4(2), 196–210.

Barlow, H. B. (1961). Possible principles unredlying the transformation of sensory messages. In W. Rosenblith (Ed.), *Sensory Communication*, pp. 217–234. Cambridge, MA: M.I.T. Press.

Barlow, H. B., T. P. Kaushal, and G. J. Mitchison (1989). Finding minimum entropy codes. *Neural Computation* 1(3), 412–423.

Jaynes, E. T. (1982). On the rationale of maximum-entropy methods. *Proc. IEEE 70*, 939–952.

Lee, D. D. and H. S. Seung (1999). Learning the parts of objects by non-negative matrix factorization. *Nature 401*(6755), 788–791.

Linsker, R. (1988). Self-organization in a perceptual network. *Computer 21*, 105–117.

Mel, B. W. (1999). Think positive to find parts. *Nature 401*(6755), 759–760.

Moghaddam, B. and A. Pentland (1997). Probabilistic visual learning for object representation. *IEEE Trans. on Pattern Analysis and Machine Intelligence 19*(7), 669–710.

Penev, P. S. (1998). *Local Feature Analysis: A Statistical Theory for Information Representation and Transmission*. Ph. D. thesis, The Rockefeller University, New York, NY. available at http://venezia.rockefeller.edu/penev/thesis/.

Penev, P. S. (1999). Dimensionality reduction by sparsification in a local-features representation of human faces. Technical report, The Rockefeller University. ftp://venezia.rockefeller.edu/pubs/PenevPS-NIPS99-reduce.ps.

Penev, P. S. and J. J. Atick (1996). Local Feature Analysis: A general statistical theory for object representation. *Network: Comput. Neural Syst. 7*(3), 477–500.

Penev, P. S. and L. G. Iordanov (1999). Local Feature Analysis: A flexible statistical framework for dimensionality reduction by sparsification of naturalistic sound. Technical report, The Rockefeller University. ftp://venezia.rockefeller.edu/pubs/PenevPS-ICASSP2000-sparse.ps.

Penev, P. S. and L. Sirovich (2000). The global dimensionality of face space. In *Proc. 4th Int'l Conf. Automatic Face and Gesture Recognition*, Grenoble, France, pp. 264–270. IEEE CS.

Poggio, T. and F. Girosi (1998). A sparse representation for function approximation. *Neural Comput. 10*(6), 1445–1454.

Ruderman, D. L. and W. Bialek (1994). Statistics of natural images: Scaling in the woods. *Phys. Rev. Lett. 73*(6), 814–817.

Shannon, C. E. (1948). A mathematical theory of communication. *Bell System Tech. J. 27*, 379–423, 623–656.

Sirovich, L. and M. Kirby (1987). Low-dimensional procedure for the characterization of human faces. *J. Opt. Soc. Am. A 4*, 519–524.
